# Learning Bounded Treewidth Bayesian Networks

**Gal Elidan**
Department of Statistics
Hebrew University
Jerusalem, 91905, Israel
galel@huji.ac.il

**Stephen Gould**
Department of Electrical Engineering
Stanford University
Stanford, CA 94305, USA
sgould@stanford.edu

## Abstract

With the increased availability of data for complex domains, it is desirable to learn Bayesian network structures that are sufficiently expressive for generalization while also allowing for tractable inference. While the method of thin junction trees can, in principle, be used for this purpose, its fully greedy nature makes it prone to overfitting, particularly when data is scarce. In this work we present a novel method for learning Bayesian networks of bounded treewidth that employs global structure modifications and that is polynomial in the size of the graph and the treewidth bound. At the heart of our method is a triangulated graph that we dynamically update in a way that facilitates the addition of chain structures that increase the bound on the model's treewidth by at most one. We demonstrate the effectiveness of our "treewidth-friendly" method on several real-life datasets. Importantly, we also show that by using global operators, we are able to achieve better generalization even when learning Bayesian networks of unbounded treewidth.

## 1 Introduction

Recent years have seen a surge of readily available data for complex and varied domains. Accordingly, increased attention has been directed towards the automatic learning of complex probabilistic graphical models [22], and in particular learning the *structure* of a Bayesian network. With the goal of making predictions or providing probabilistic explanations, it is desirable to learn models that generalize well and at the same time have low inference complexity or a small treewidth [23].

While learning optimal tree-structured models is easy [5], learning the optimal structure of general and even quite simple (e.g., poly-trees, chains) Bayesian networks is computationally difficult [8, 10, 19]. Several works attempt to generalize the tree-structure result of Chow and Liu [5], either by making assumptions about the true distribution (e.g., [1, 21]), by searching for a local maxima over tree mixtures [20], or by approximate methods that are polynomial in the size of the graph but exponential in the treewidth bound (e.g., [3, 15]). In the context of general Bayesian networks, the thin junction tree approach of Bach and Jordan [2] is a local greedy search procedure that relies at each step on tree-decomposition heuristic techniques for computing an upper bound the true treewidth of the model. Like any local search approach, this method does not provide performance guarantees but is appealing in its ability to efficiently learn models with an arbitrary treewidth bound.

The thin junction tree method, however, suffers from two important limitations. First, while useful on average, even the best of the tree-decomposition heuristics exhibit some variance in the treewidth estimate [16]. As a result, a single edge addition can lead to a jump in the *treewidth estimate* despite the fact that it can increase the *true treewidth* by at most one. More importantly, structure learning scores (e.g., BIC, BDe) tend to learn spurious edges that result in overfitting when the number of samples is relatively small, a phenomenon that is made worse by a fully greedy approach. Intuitively, to generalize well, we want to learn bounded treewidth Bayesian networks where structure modifications are globally beneficial (i.e., contribute to the score in many regions of the network).

In this work we propose a novel method for efficiently learning Bayesian networks of bounded treewidth that addresses these concerns. At the heart of our method is a dynamic update of the triangulation of the model in a way that is tree-width friendly: the treewidth of the triangulated graph (upper bound on the model's true treewidth) is guaranteed to increase by at most one when an

edge is added to the network. Building on the single edge triangulation, we characterize sets of edges that are *jointly* treewidth-friendly. We use this characterization in a dynamic programming approach for learning the optimal treewidth-friendly chain with respect to a node ordering. Finally, we learn a bounded treewidth Bayesian network by iteratively augmenting the model with such chains.

Instead of using local edge modifications, our method progresses by incrementally adding chain structures that are globally beneficial, improving our ability to generalize. We are also able to *guarantee* that the bound on the model's treewidth grows by at most one at each iteration. Thus, our method resembles the global nature of Chow and Liu [5] more closely than the thin junction tree approach of Bach and Jordan [2], while being applicable in practice to any desired treewidth.

We evaluate our method on several challenging real-life datasets and show that our method is able to learn richer models that generalize better than the thin junction tree approach as well as an unbounded aggressive search strategy. Furthermore, we show that even when learning models with unbounded treewidth, by using global structure modification operators, we are better able to cope with the problem of local maxima and learn better models.

## 2 Background: Bayesian networks and tree decompositions

A *Bayesian network* [22] is a pair $(\mathcal{G}, \Theta)$ that encodes a joint probability distribution over a finite set $\mathcal{X} = \{X_1, \ldots, X_n\}$ of random variables. $\mathcal{G}$ is a directed acyclic graph whose nodes correspond to the variables in $\mathcal{X}$. The parameters $\Theta_{X_i \mid \mathbf{Pa}_i}$ encode local *conditional probability distribution*s (CPDs) for each node $X_i$ given its parents in $\mathcal{G}$. Together, these define a unique joint probability distribution over $\mathcal{X}$ given by $P(X_1, \ldots, X_n) = \prod_{i=1}^{n} P(X_i \mid \mathbf{Pa}_i)$.

Given a structure $\mathcal{G}$ and a complete training set $\mathcal{D}$, estimating the (regularized) *maximum likelihood* (ML) parameters is easy for many choices of CPDs (see [14] for details). Learning the structure of a network, however, is generally NP-hard [4, 10, 19] as the number of possible structures is super-exponential in the number of variables. In practice, structure learning relies on a greedy search procedure that examines easy to evaluate local structure changes (add, delete or reverse an edge). This search is usually guided by a decomposable score that balances the likelihood of the data and the complexity of the model (e.g., BIC [24], *Bayesian score* [14]). Chow and Liu [5] showed that the ML tree can be learned efficiently. Their result is easily generalized to any decomposable score.

Given a model, we are interested in the task of inference, or evaluating queries of the form $P(\mathbf{Y} \mid \mathbf{Z})$ where $\mathbf{Y}$ and $\mathbf{Z}$ are arbitrary subsets of $\mathcal{X}$. This task is, in general, NP-hard [7], except when $\mathcal{G}$ is tree structured. The actual complexity of inference in a Bayesian network is proportional to its *treewidth* [23] which, roughly speaking, measures how closely the network resembles a tree. The notions of tree-decompositions and treewidth were introduced by Robertson and Seymour [23]:[1]

**Definition 2.1:** A tree-decomposition of an undirected graph $\mathcal{H} = (\mathbf{V}, \mathbf{E})$ is a pair $(\{\mathbf{C}_i\}_{i \in \mathcal{T}}, \mathcal{T})$, where $\mathcal{T}$ is a tree, $\{\mathbf{C}_i\}$ is a subset of $\mathbf{V}$ such that $\bigcup_{i \in \mathcal{T}} \mathbf{C}_i = \mathbf{V}$ and where
- for all edges $(v, w) \in \mathbf{E}$ there exists an $i \in \mathcal{T}$ with $v \in \mathbf{C}_i$ and $w \in \mathbf{C}_i$.
- for all $i, j, k \in \mathcal{T}$: if $j$ is on the (unique) path from $i$ to $k$ in $\mathcal{T}$, then $\mathbf{C}_i \cap \mathbf{C}_k \subseteq \mathbf{C}_j$. ∎

The treewidth of a tree-decomposition is defined to be $\max_{i \in \mathcal{T}} |\mathbf{C}_i| - 1$. The treewidth $TW(\mathcal{H})$ of an undirected graph $\mathcal{H}$ is the minimum treewidth over all possible tree-decompositions of $\mathcal{H}$. An equivalent notion of treewidth can be phrased in terms of a graph that is a triangulation of $\mathcal{H}$.

**Definition 2.2:** An induced path $\mathcal{P}$ in an undirected graph $\mathcal{H}$ is a path such that for every non-adjacent vertices $p_i, p_j \in \mathcal{P}$ there is no edge $(p_i—p_j) \in \mathcal{H}$. A triangulated (chordal) graph is an undirected graph with no induced cycles. Equivalently, it is an undirected graph in which every cycle of length four or more contains a chord. ∎

It can be easily shown that the treewidth of a triangulated graph is the size of the maximal clique of the graph minus one [23]. The treewidth of an undirected graph $\mathcal{H}$ is then the minimum treewidth of all triangulations of $\mathcal{H}$. For the underlying directed acyclic graph of a Bayesian network, the treewidth can be characterized via a triangulation of the moralized graph.

**Definition 2.3:** A moralized graph $\mathcal{M}$ of a directed acyclic graph $\mathcal{G}$ is an undirected graph that has an edge $(i—j)$ for every $(i \rightarrow j) \in \mathcal{G}$ and an edge $(p—q)$ for every pair $(p \rightarrow i), (q \rightarrow i) \in \mathcal{G}$. ∎

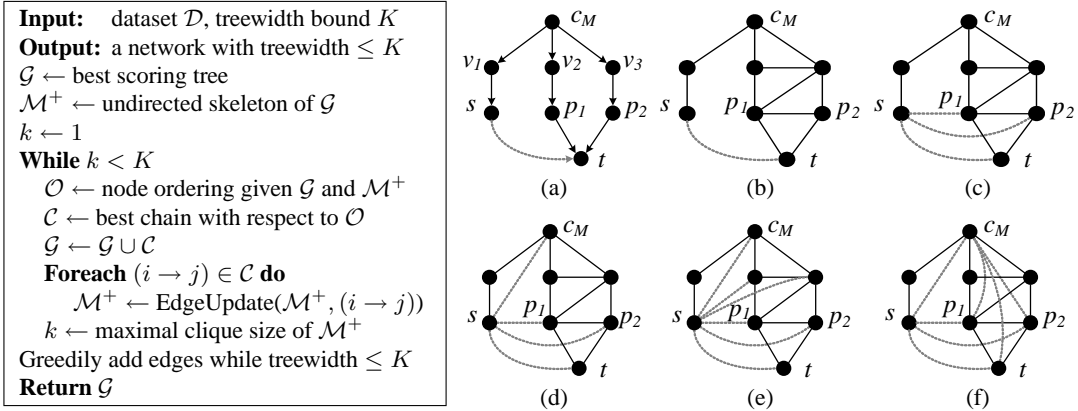

**Input:** dataset $\mathcal{D}$, treewidth bound $K$
**Output:** a network with treewidth $\leq K$
$\mathcal{G} \leftarrow$ best scoring tree
$\mathcal{M}^+ \leftarrow$ undirected skeleton of $\mathcal{G}$
$k \leftarrow 1$
**While** $k < K$
   $\mathcal{O} \leftarrow$ node ordering given $\mathcal{G}$ and $\mathcal{M}^+$
   $\mathcal{C} \leftarrow$ best chain with respect to $\mathcal{O}$
   $\mathcal{G} \leftarrow \mathcal{G} \cup \mathcal{C}$
   **Foreach** $(i \rightarrow j) \in \mathcal{C}$ **do**
      $\mathcal{M}^+ \leftarrow$ EdgeUpdate$(\mathcal{M}^+, (i \rightarrow j))$
   $k \leftarrow$ maximal clique size of $\mathcal{M}^+$
Greedily add edges while treewidth $\leq K$
**Return** $\mathcal{G}$

(a)      (b)      (c)

(d)      (e)      (f)

**Figure 1: (left)** Outline of our algorithm for learning Bayesian networks of bounded treewidth. **(right)** An example of the different steps of our triangulation procedure (b)-(e) when $(s \rightarrow t)$ is added to the graph in (a). The blocks are $\{s, v_1\}$, $\{v_1, c_M\}$, and $\{c_M, v_2, v_3, p_1, p_2, t\}$ with corresponding cut-vertices $v_1$ and $c_M$. The augmented graph (e) has a treewidth of three (maximal clique of size four). An alternative triangulation (f), connecting $c_M$ to $t$, would result in a maximal clique of size five.

The treewidth of a Bayesian network graph $\mathcal{G}$ is defined as the treewidth of its moralized graph $\mathcal{M}$. It follows that the maximal clique of *any* moralized triangulation of $\mathcal{G}$ is an upper bound on the treewidth of the model, and thus its inference complexity.

## 3 Learning Bounded Treewidth Bayesian Networks

In this section we outline our approach for learning Bayesian networks given an arbitrary treewidth bound that is polynomial in both the number of variables and the desired treewidth. We rely on global structure modifications that are optimal with respect to a node ordering.

At the heart of our method is the idea of using a dynamically maintained triangulated graph to upper bound the treewidth of the current model. When an edge is added to the Bayesian network we update this triangulated graph in a way that is not only guaranteed to produce a valid triangulation, but that is also treewidth-friendly. That is, our update is guaranteed to increase the size of the maximal clique of the triangulated graph, and hence the treewidth bound, by at most one. An important property of our edge update is that we can characterize the parts of the network that are "contaminated" by the new edge. This allows us to define sets of edges that are *jointly* treewidth-friendly. Building on the characterization of these sets, we propose a dynamic programming approach for efficiently learning the optimal treewidth-friendly chain with respect to a node ordering.

Figure 1 shows pseudo-code for our method. Briefly, we learn a Bayesian network with bounded treewidth $K$ by starting from a Chow-Liu tree and iteratively augmenting the current structure with an optimal *treewidth-friendly chain*. During each iteration (below the treewidth bound) we apply our treewidth-friendly edge update procedure that maintains a moralized and triangulated graph for the model at hand. Appealingly, as each global modification can increase the treewidth by at most one, at least $K$ such chains will be added before we face the problem of local maxima. In practice, as some chains do not increase the treewidth, many more such chains are added for a given $K$.

**Theorem 3.1:** Given a treewidth bound $K$ and dataset over $N$ variables, the algorithm outlined in Figure 1 runs in time polynomial in $N$ and $K$.

This result relies on the efficiency of each step of the algorithm and that there can be at most $N \cdot K$ iterations ($\leq$ |edges|) before exceeding the treewidth bound. In the next sections we develop the edge update and best scoring chain procedures and show that both are polynomial in $N$ and $K$.

## 4 Treewidth-Friendly Edge Update

The basic building block of our method is a procedure for maintaining a valid triangulation of the Bayesian network. An appealing feature of this procedure is that the treewidth bound is guaranteed to grow by at most one after the update. We first consider single edge $(s \rightarrow t)$ addition to the model. For clarity of exposition, we start with a simple variant of our procedure, and later refine this to allow for multiple edge additions while maintaining our guarantee on the treewidth bound.

To gain intuition into how the dynamic nature of our update is useful, we use the notion of induced paths or paths with no shortcuts (see Section 2), and make explicit the following obvious fact:

**Observation 4.1:** Let $\mathcal{G}$ be a Bayesian network structure and let $\mathcal{M}^+$ be a moralized triangulation of $\mathcal{G}$. Let $\mathcal{M}_{(s \to t)}$ be $\mathcal{M}^+$ augmented with the edge $(s\!-\!t)$ and with the edges $(s\!-\!p)$ for every parent $p$ of $t$ in $\mathcal{G}$. Then, every non-chordal cycle in $\mathcal{M}_{(s \to t)}$ involves $s$ and either $t$ or a parent of $t$ and an induced path between the two vertices. ∎

Stated simply, if the graph was triangulated before the addition of $(s \to t)$ to the Bayesian network, then we only need to triangulate cycles created by the addition of the new edge or those forced by moralization. This observation immediately suggests a straight-forward *single-source triangulation* whereby we simply add an edge $(s\!-\!v)$ for every node $v$ on an induced path between $s$ and $t$ or its parents before the edge update. Clearly, this naive method results in a valid moralized triangulation of $\mathcal{G} \cup (s \to t)$. Surprisingly, we can also show that it is treewidth-friendly.

**Theorem 4.2:** The treewidth of the graph produced by the *single-source triangulation* procedure is greater than the treewidth of the input graph $\mathcal{M}^+$ by at most one.

**Proof:** (**outline**) For the treewidth to increase by more than one, some maximal $\mathbf{C}$ in $\mathcal{M}^+$ needs to connect to two new nodes. Since all edges are being added from $s$, this can only happen in one of two ways: (i) either $t$, a parent $p$ of $t$, or a node $v$ on induced path between $s$ and $t$ is also connected to $\mathbf{C}$, but not part of $\mathbf{C}$, or (ii) two such (non-adjacent) nodes exist and $s$ is in $\mathbf{C}$. In either case one edge is missing after the update procedure preventing the formation of a larger clique. ∎

One problem with the proposed single-source triangulation, despite it being treewidth-friendly, is that many vertices are connected to the source node, making the triangulations shallow. This can have an undesirable effect on future edge additions and increases the chances of the formation of large cliques. We can alleviate this problem with a refinement of the single-source triangulation procedure that makes use of the concepts of cut-vertices, blocks, and block trees.

**Definition 4.3:** A block of an undirected graph $\mathcal{H}$ is a set of connected nodes that cannot be disconnected by the removal of a single vertex. By convention, if the edge $(u\!-\!v)$ is in $\mathcal{H}$ then $u$ and $v$ are in the same block. Vertices that separate (are in the intersection of) blocks are called cut-vertices. ∎

It is easy to see that between every two nodes in a block of size greater than two there are at least two distinct paths, i.e. a cycle. There are also no simple cycles involving nodes in different blocks.

**Definition 4.4:** The (unique) block tree $\mathcal{B}$ of an undirected graph $\mathcal{H}$ is a graph with nodes that correspond both to cut-vertices and to blocks of $\mathcal{H}$. The edges in the block tree connect any block node $\mathbf{B}_i$ with a cut-vertex node $v_j$ if and only if $v_j \in \mathbf{B}_i$ in $\mathcal{H}$. ∎

It can be easily shown that any path in $\mathcal{H}$ between two nodes in different blocks passes through all the cut-vertices along the path between the blocks in $\mathcal{B}$. An important consequence that follows from Dirac [11] is that an undirected graph whose blocks are triangulated is overall triangulated.

Our refined treewidth-friendly triangulation procedure (illustrated via an example in Figure 1) makes use of this fact as follows. First, the triangulated graph is augmented with the edge $(s\!-\!t)$ and any edges needed for moralization (Figure 1(b) and (c)). Second, a block level triangulation is carried out by zig-zagging across cut-vertices along the unique path between the blocks containing $s$ and $t$ and its parents (Figure 1(d)). Next, within each block (not containing $s$ or $t$) along the path, a single-source triangulation is performed with respect to the "entry" and "exit" cut-vertices. This short-circuits any other *node path* through (and within) the block. For the block containing $s$ the single-source triangulation is performed between $s$ and the "exit" cut-vertex. The block containing $t$ and its parents is treated differently: we add chords directly from $s$ to any node $v$ within the block that is on an *induced path* between $s$ and $t$ (or parents of $t$) (Figure 1(e)). This is required to prevent moralization and triangulation edges from interacting in a way that will increase the treewidth by more than one (e.g., Figure 1(f)). If $s$ and $t$ happen to be in the same block, then we only triangulate the induced paths between $s$ and $t$, i.e., the last step outlined above. Finally, in the special case that $s$ and $t$ are in *disconnected* components of $\mathcal{G}$, the only edges added are those required for moralization.

**Theorem 4.5:** Our revised edge update procedure results in a triangulated graph with a treewidth at most one greater than that of the input graph. Furthermore, it runs in polynomial time.

**Proof:** (**outline**) First, observe that the final step of adding chords emanating from $s$ is a single-source triangulation once the other steps have been performed. Since each block along the block path between $s$ and $t$ is triangulated separately, we only need to consider the zig-zag triangulation between blocks. As this creates 3-cycles, the graph must also be triangulated. To see that the treewidth

increases by at most one, we use similar arguments to those used in the proof of Theorem 4.2, and observe that the zig-zag triangulation only touches cut-vertices and any three of these vertices could not have been in the same clique. The fact that the update procedure runs in polynomial time follows from the fact that an adaptation (not shown for lack of space) of maximum cardinality search (see, for example [16]) can be used to efficiently identify all induced nodes between $s$ and $t$. ∎

**Multiple Edge Updates.** We now consider the addition of multiple edges to the graph $\mathcal{G}$. To ensure that multiple edges do not interact in ways that will increase the treewidth bound by more than one, we need to characterize the nodes *contaminated* by each edge addition—a node $v$ is contaminated by the adding $(s \rightarrow t)$ to $\mathcal{G}$ if it is incident to a *new* edge added during our treewidth friendly triangulation. Below are several examples of contaminated sets (solid nodes) incident to edges added (dashed) by our edge update procedure for different candidate edge additions $(s \rightarrow t)$ to the Bayesian network on the left. In all examples except the last treewidth is increased by one.

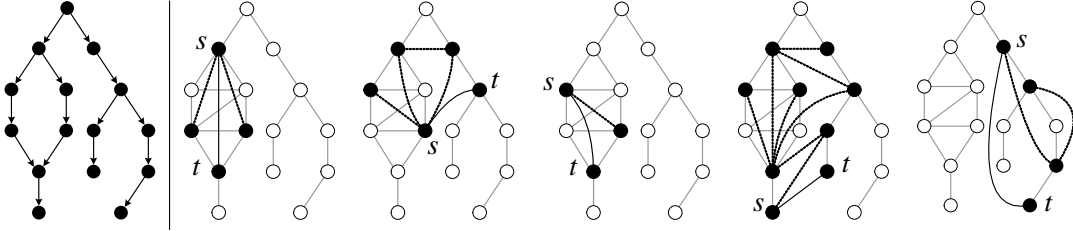

Using the notion of contamination, we can characterize sets of edges that are *jointly* treewidth-friendly. We will use this to learn optimal treewidth friendly chains given a ordering in Section 5.

**Theorem 4.6: (Treewidth-friendly set).** Let $\mathcal{G}$ be a graph structure and $\mathcal{M}^+$ be its corresponding moralized triangulation. If $\{(s_i \rightarrow t_i)\}$ is a set of candidate edges satisfying the following:

- the contaminated sets of any $(s_i \rightarrow t_i)$ and $(s_j \rightarrow t_j)$ are disjoint, or,
- the contaminated sets overlap at a single cut-vertex, *but* the endpoints of each edge are not in the same block *and* the block paths between the endpoints do not overlap;

then adding all edges to $\mathcal{G}$ can increase the treewidth bound by at most one.

**Proof:** (**outline**) The theorem holds trivially for the first condition. Under the second condition, the only common vertex is a cut-vertex. However, since all other contaminated nodes are in in different blocks, they cannot interact to form a large clique. ∎

# 5 Learning Optimal Treewidth-Friendly Chains

In the previous section we described our edge update procedure and characterized edge chains that jointly increase the treewidth bound by at most one. We now use this to search for optimal chain structures that satisfy Theorem 4.6, and are thus treewidth friendly, given a topological node ordering. On the surface, one might question the need for a specific node ordering altogether if chain global operators are to be used—given the result of Chow and Liu [5], one might expect that learning the optimal chain with respect to *any* ordering can be carried out efficiently. However, Meek [19] showed that learning an optimal chain over a set of random variables is computationally difficult and the result can be generalized to learning a chain conditioned the current model. Thus, during any iteration of our algorithm, we cannot expect to find the overall optimal chain.

Instead, we commit to a single node ordering that is topologically consistent (each node appears after its parent in the network) and learn the optimal treewidth-friendly chain with respect to that order (we briefly discuss the details of our ordering below). To find such a chain in polynomial time, we use a straightforward dynamic programming approach: the best treewidth-friendly chain that contains $(\mathcal{O}_s \rightarrow \mathcal{O}_t)$ is the concatenation of:

- the best chain from the first node $\mathcal{O}_1$ to $\mathcal{O}_F$, the first node contaminated by $(\mathcal{O}_s \rightarrow \mathcal{O}_t)$
- the edge $(\mathcal{O}_s \rightarrow \mathcal{O}_t)$
- the best chain starting from the last node contaminated $\mathcal{O}_L$ to the last node in the order $\mathcal{O}_N$.

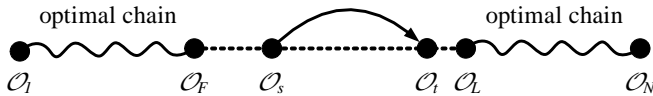

We note that when the end nodes are not separating cut-vertices, we maintain a gap so that the contamination sets are disjoint and the conditions of Theorem 4.6 are met.

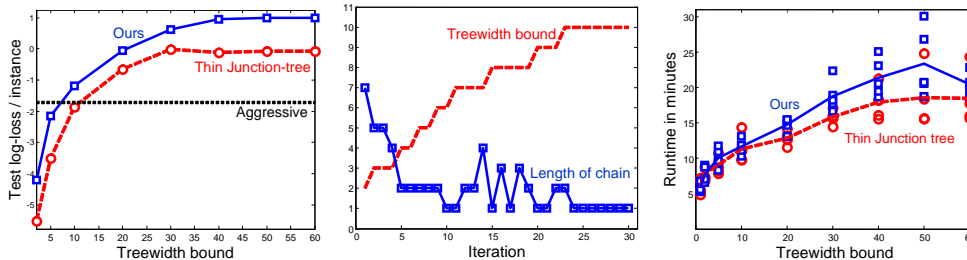

**Figure 2:** Gene expression results: **(left)** 5-fold mean test log-loss per/instance vs. treewidth bound. Our method (solid blue squares) is compared to the thin junction tree method (dashed red circles), and an unbounded aggressive search (dotted black). **(middle)** the treewidth estimate and the number of edges in the chain during the iterations of a typical run with the bound set to 10. **(right)** shows running time as a function of the bound.

Formally, we define $C[i, j]$ as the optimal chain whose contamination is limited to the range $[\mathcal{O}_i, \mathcal{O}_j]$ and our goal is to compute $C[1, N]$. Using F to denote the first node ordered in the contamination set of $(s \rightarrow t)$ (and L for the last), we can compute $C[1, N]$ via the following recursive update principle

$$C[i, j] = \begin{cases} \max_{s,t:F=i,L=j}(s \rightarrow t) & \text{no split} \\ \max_{k=i+1:j-1} C[i, k] \cup C[k, j] & \text{split} \\ \emptyset & \text{leave a gap} \end{cases}$$

where the maximization is with respect to the structure score (e.g., BIC). That is, the best chain in a subsequence $[i, j]$ in the ordering is the maximum of three alternatives: edges whose contamination boundaries are exactly $i$ and $j$ (no split); two chains that are joined at some node $i < k < j$ (split); a gap between $i$ and $j$ when there is no positive edge whose contamination is in $[i, j]$.

Finally, for lack of space we only provide a brief description of our topological node ordering. Intuitively, since edges contaminate nodes along the block path between the edge's endpoints (see Section 4), we want to adopt a DFS ordering over the blocks so as to facilitate as many edges as possible between different branches of the block tree. We order nodes with a block by the distance from the "entry" vertex as motivated by the following result on the distance $d_{\min}^M(u, v)$ between nodes $u, v$ in the triangulated graph $\mathcal{M}^+$ (proof not shown for lack of space):

**Theorem 5.1:** Let $r, s, t$ be nodes in a block $\mathbf{B}$ in the triangulated graph $\mathcal{M}^+$ with $d_{\min}^M(r, s) \leq d_{\min}^M(r, t)$. Then for any $v$ on an induced path between $s$ and $t$ we have $d_{\min}^M(r, v) \leq d_{\min}^M(r, t)$.

The efficiency of our method outlined in Figure 1 in the number of variables and the treewidth bound (Theorem 3.1) now follows from the efficiency of the ordering and chain learning procedures.

# 6 Experimental Evaluation

We compare our approach on four real-world datasets to several methods. The first is an improved variant of the thin junction tree method [2]. We start (as in our method) with a Chow-Liu forest and iteratively add the single best scoring edge as long as the treewidth bound is not exceeded. To make the comparison independent of the choice of triangulation method, at each iteration we replace the heuristic triangulation (best of maximum cardinality search or minimum fill-in [16], which in practice had negligible differences) with our triangulation if it results in a lower treewidth.The second baseline is an aggressive structure learning approach that combines greedy edge modifications with a TABU list (e.g., [13]) and random moves and that is not constrained by a treewidth bound. Where relevant we also compare our results to the results of Chechetka and Guestrin [3].

**Gene Expression.** We first consider a continuous dataset of the expression of yeast genes (variables) in 173 experiments (instances) [12]. We learn sigmoid Bayesian networks using the BIC structure score [24] using the fully observed set of 89 genes that participate in general metabolic processes. Here a learned model indicates possible regulatory or functional connections between genes.

Figure 2(a) shows test log-loss as a function of treewidth bound. The first obvious phenomenon is that both our method and the thin junction tree approach are superior to the aggressive baseline. As one might expect, the aggressive baseline achieves a higher BIC score on training data (not shown), but overfits due to the scarcity of the data. The consistent superiority of our method over thin junction trees demonstrates that a better choice of edges, i.e., ones chosen by a global operator, can lead to increased robustness and better generalization. Indeed, even when the treewidth bound

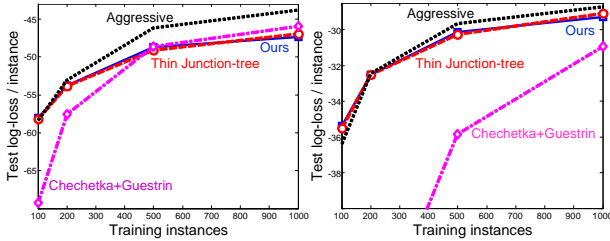
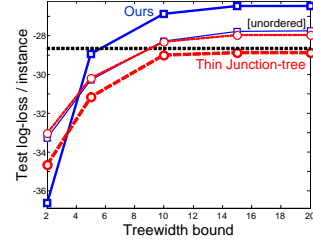

**Figure 3:** 5-fold mean test log-loss/instance for a treewidth bound of two vs. training set size for the temperature (left) and traffic (right) datasets. Compared are our approach (solid blue squares), the thin junction tree method (dashed red circles), an aggressive unbounded search (dotted black), and the method of Chechetka and Guestrin [3] (dash-dot magenta diamonds).

**Figure 4:** Average log-loss vs. treewidth bound for the Hapmap data. Compared are an unbounded aggressive search (dotted) and unconstrained (thin) and constrained by the DNA order (thick) variants of ours and the thin junction tree method.

is increased past the saturation point, our method surpasses both baselines. In this case, we are learning unbounded networks and all benefit comes from the global nature of our updates.

To qualitatively illustrate the progression of our algorithm, in Figure 2(b) we plot the number of edges in the chain and the treewidth estimate at the end of each iteration for a typical run. Our algorithm aggressively adds multi-edge chains until the treewidth bound is reached, at which point (iteration 24) it becomes fully greedy. To appreciate the non-triviality of some of the chains learned with $4-7$ edges, we recall that the chains are added *after* a Chow-Liu model was initially learned. It is also worth noting that despite their complexity, some chains do not increase the treewidth estimate and we typically have more than $K$ iterations where chains with more than one edge are added. The number of such iterations is still polynomially bounded as for a Bayesian network with $N$ variables adding more than $K \cdot N$ edges will necessarily result in a treewidth that is greater than $K$.

To evaluate the efficiency of our method we measured its running time as a function of the treewidth bound. Figure 2(c) shows results for the gene expression dataset. Observe that our method and the greedy thin junction tree approach are both approximately linear in the treewidth bound. Appealingly, the additional computation our method requires is not significant ($\leq 25\%$). This should not come as a surprise as the bulk of the time is spent on the collection of the data sufficient statistics.

It is also worth discussing the range of treewidths we considered in the above experiment as well as the Haplotype experiment below. While treewidths greater than 25 seem excessive for exact inference, state-of-the-art techniques (e.g., [9, 18]) can reasonably handle inference in networks of this complexity. Furthermore, as our results show, it is beneficial in practice to learn such models. Thus, combining our method with state-of-the-art inference techniques can allow practitioners to push the envelope of the complexity of models learned for real applications that rely on exact inference.

**The Traffic and Temperature Datasets.** We now compare our method to the mutual-information based LPACJT approach of Chechetka and Guestrin [3] (we compare to the better variant). As their method is exponential in the treewidth and cannot be used in the gene expression setting, we compare to it on the two discrete real-life datasets Chechetka and Guestrin [3] considered: the temperature data is from a deployment of 54 sensor nodes; the traffic dataset contains traffic flow information measured every 5 minutes in 32 locations in California. To make the comparison fair, we used the same discretization and train/test splits. Furthermore, as their method can only be applied to a small treewidth bound, we also limited our model to a treewidth of two. Figure 3 compares the different methods. Both our method and the thin junction tree approach significantly outperform the LPACJT on small sample sizes. This result is consistent with the results reported in Chechetka and Guestrin [3] and is due to the fact that the LPACJT method does not facilitate the use of regularization which is crucial in the sparse-data regime. The performance of our method is comparable to the greedy thin junction tree approach with no obvious superiority of either method. This should not come as a surprise since the fact that the unbounded aggressive search is not significantly better suggests that the strong signal in the data can be captured rather easily. In fact, Chechetka and Guestrin [3] show that even a Chow-Liu tree does rather well on these datasets (compare this to the gene expression dataset where the aggressive variant was superior even at a treewidth of five).

**Haplotype Sequences.** Finally we consider a more difficult discrete dataset of a sequence of single nucleotide polymorphism (SNP) alleles from the Human HapMap project [6]. Our model is defined over 200 SNPs (binary variables) from chromosome 22 of a European population consisting of 60 individuals (we considered several different sequences along the chromosome with similar results).

In this case, there is a natural ordering of variables that corresponds to the position of the SNPs in the DNA sequence. Figure 4 shows test log-loss results when this ordering is enforced (thicker) and when it is not (thinner). The superiority of our method when the ordering is used is obvious while the performance of the thin junction tree method degrades. This can be expected as the greedy method does not make use of a node ordering, while our method provides optimality guarantees with respect to a variable ordering at each iteration. Whether constrained to the natural variable ordering or not, our method ultimately also surpasses the unbounded aggressive search.

## 7   Discussion and Future Work

In this work we presented a novel method for learning Bayesian networks of bounded treewidth in time that is polynomial in *both* the number of variables and the treewidth bound. Our method builds on an edge update algorithm that dynamically maintains a valid moralized triangulation in a way that facilitates the addition of chains that are guaranteed to increase the treewidth bound by at most one. We demonstrated the effectiveness of our treewidth-friendly method on real-life datasets, and showed that by utilizing global structure modification operators, we are able to learn better models than competing methods, even when the treewidth of the models learned is not constrained.

Our method can be viewed as a generalization of the work of Chow and Liu [5] that is constrained to a chain structure but that provides an optimality guarantee (with respect to a node ordering) at every treewidth. In addition, unlike the thin junction trees approach of Bach and Jordan [2], we provide a guarantee that our estimate of the treewidth bound will not increase by more than one at each iteration. Furthermore, we add multiple edges at each iteration, which in turn allows us to better cope with the problem of local maxima in the search. To our knowledge, ours is the first method for efficiently learning Bayesian networks with an arbitrary treewidth bound that is not fully greedy.

Our method motivates several exciting future directions. It would be interesting to see to what extent we could overcome the need to commit to a specific node ordering at each iteration. While we provably cannot consider every ordering, it may be possible to polynomially provide a reasonable approximation. Second, it may be possible to refine our characterization of the contamination that results from an edge update, which in turn may facilitate the addition of more complex treewidth-friendly structures at each iteration. Finally, we are most interested in exploring whether tools similar to the ones employed in this work could be used to dynamically update the bounded treewidth structure that is the approximating distribution in a variational approximate inference setting.

## Footnotes

[1] The tree-decomposition properties are equivalent to the corresponding *family preserving* and *running intersection* properties of clique trees introduced by Lauritzen and Spiegelhalter [17] at around the same time.

## References

[1] P. Abbeel, D. Koller, and A. Ng. Learning factor graphs in poly. time & sample complexity. *JMLR*, 2006.

[2] F. Bach and M. I. Jordan. Thin junction trees. In *NIPS*, 2001.

[3] A. Chechetka and C. Guestrin. Efficient principled learning of thin junction trees. In *NIPS*. 2008.

[4] D. Chickering. Learning Bayesian networks is NP-complete. In *Learning from Data: AI & Stats V*. 1996.

[5] C. Chow and C. Liu. Approx. discrete distrib. with dependence trees. *IEEE Trans. on Info. Theory*, 1968.

[6] The International HapMap Consortium. The international hapmap project. *Nature*, 2003.

[7] G. F. Cooper. The computationl complexity of probabilistic inference using belief networks. *AI*, 1990.

[8] P. Dagum and M. Luby. An optimal approximation algorithm for baysian inference. *AI*, 1993.

[9] A. Darwiche. Recursive conditioning. *Artificial Intelligence*, 2001.

[10] S. Dasgupta. Learning polytrees. In *UAI*, 1999.

[11] G. A. Dirac. On rigid circuit graphs. Abhandlungen aus dem Math. Seminar der Univ. Hamburg 25, 1961.

[12] A. Gasch et al. Genomic expression program in the response of yeast cells to environmental changes. *Molecular Biology of the Cell*, 2000.

[13] F. Glover and M. Laguna. Tabu search. In *Modern Heuristic Tech. for Comb. Problems*, 1993.

[14] D. Heckerman. A tutorial on learning Bayesian networks. Technical report, Microsoft Research, 1995.

[15] D. Karger and N. Srebro. Learning markov networks: maximum bounded tree-width graphs. In *Symposium on Discrete Algorithms*, 2001.

[16] A. Koster, H. Bodlaender, and S. Van Hoesel. Treewidth: Computational experiments. Technical report, Universiteit Utrecht, 2001.

[17] S. Lauritzen and D. Spiegelhalter. Local computations with probabilities on graphical structures. *Journal of the Royal Statistical Society*, 1988.

[18] R. Marinescu and R. Dechter. And/or branch-and-bound for graphical models. *IJCAI*, 2005.

[19] C. Meek. Finding a path is harder than finding a tree. *Journal of Artificial Intelligence Research*, 2001.

[20] M. Meila and M. I. Jordan. Learning with mixtures of trees. *JMLR*, 2000.

[21] M. Narasimhan and J. Bilmes. Pac-learning bounded tree-width graphical models. In *UAI*, 2004.

[22] J. Pearl. *Probabilistic Reasoning in Intelligent Systems*. Morgan Kaufmann, 1988.

[23] N. Robertson and P. Seymour. Graph minors II. algorithmic aspects of tree-width. *J. of Algorithms*, 1987.

[24] G. Schwarz. Estimating the dimension of a model. *Annals of Statistics*, 6:461–464, 1978.

